# Independent Factor Analysis with Temporally Structured Sources

**Hagai Attias**
hagai@gatsby.ucl.ac.uk
Gatsby Unit, University College London
17 Queen Square
London WC1N 3AR, U.K.

## Abstract

We present a new technique for time series analysis based on dynamic probabilistic networks. In this approach, the observed data are modeled in terms of unobserved, mutually independent factors, as in the recently introduced technique of Independent Factor Analysis (IFA). However, unlike in IFA, the factors are not i.i.d.; each factor has its own temporal statistical characteristics. We derive a family of EM algorithms that learn the structure of the underlying factors and their relation to the data. These algorithms perform source separation and noise reduction in an integrated manner, and demonstrate superior performance compared to IFA.

## 1 Introduction

The technique of independent factor analysis (IFA) introduced in [1] provides a tool for modeling $L'$-dim data in terms of $L$ unobserved factors. These factors are mutually independent and combine linearly with added noise to produce the observed data. Mathematically, the model is defined by

$$\mathbf{y}_t = \mathbf{H}\mathbf{x}_t + \mathbf{u}_t \,, \tag{1}$$

where $\mathbf{x}_t$ is the vector of factor activities at time $t$, $\mathbf{y}_t$ is the data vector, $\mathbf{H}$ is the $L' \times L$ mixing matrix, and $\mathbf{u}_t$ is the noise.

The origins of IFA lie in applied statistics on the one hand and in signal processing on the other hand. Its statistics ancestor is ordinary factor analysis (FA), which assumes Gaussian factors. In contrast, IFA allows each factor to have its own arbitrary distribution, modeled semi-parametrically by a 1-dim mixture of Gaussians (MOG). The MOG parameters, as well as the mixing matrix and noise covariance matrix, are learned from the observed data by an expectation-maximization (EM) algorithm derived in [1]. The signal processing ancestor of IFA is the independent component analysis (ICA) method for blind source separation [2]–[6]. In ICA, the factors are termed *sources*, and the task of blind source separation is to recover them from the observed data with no knowledge of the mixing process. The sources in ICA have non-Gaussian distributions, but unlike in IFA these distributions are usually fixed by prior knowledge or have quite limited adaptability. More significant restrictions

are that their number is set to the data dimensionality, i.e. $L = L'$ ('square mixing'), the mixing matrix is assumed invertible, and the data are assumed noise-free ($\mathbf{u}_t = 0$). In contrast, IFA allows any $L, L'$ (including more sources than sensors, $L > L'$), as well as non-zero noise with unknown covariance. In addition, its use of the flexible MOG model often proves crucial for achieving successful separation [1].

Therefore, IFA generalizes and unifies FA and ICA. Once the model has been learned, it can be used for classification (fitting an IFA model for each class), completing missing data, and so on. In the context of blind separation, an optimal reconstruction of the sources $\mathbf{x}_t$ from data is obtained [1] using a MAP estimator.

However, IFA and its ancestors suffer from the following shortcoming: They are oblivious to temporal information since they do not attempt to model the temporal statistics of the data (but see [4] for square, noise-free mixing). In other words, the model learned would not be affected by permuting the time indices of $\{\mathbf{y}_t\}$. This is unfortunate since modeling the data as a time series would facilitate filtering and forecasting, as well as more accurate classification. Moreover, for source separation applications, learning temporal statistics would provide additional information on the sources, leading to cleaner source reconstructions.

To see this, one may think of the problem of blind separation of noisy data in terms of two components: source separation and noise reduction. A possible approach might be the following two-stage procedure. First, perform noise reduction using, e.g., Wiener filtering. Second, perform source separation on the cleaned data using, e.g., an ICA algorithm. Notice that this procedure directly exploits temporal (second-order) statistics of the data in the first stage to achieve stronger noise reduction. An alternative approach would be to exploit the temporal structure of the data indirectly, by using a temporal source model. In the resulting single-stage algorithm, *the operations of source separation and noise reduction are coupled.* This is the approach taken in the present paper.

In the following, we present a new approach to the independent factor problem based on dynamic probabilistic networks. In order to capture temporal statistical properties of the observed data, we describe each source by a hidden Markov model (HMM). The resulting dynamic model describes a multivariate time series in terms of several independent sources, each having its own temporal characteristics. Section 2 presents an EM learning algorithm for the zero-noise case, and section 3 presents an algorithm for the case of isotropic noise. The case of non-isotropic noise turns out to be computationally intractable; section 4 provides an approximate EM algorithm based on a variational approach.

**Notation:** The multivariable Gaussian density is denoted by $\mathcal{G}(\mathbf{z}, \boldsymbol{\Sigma}) = |\, 2\pi\boldsymbol{\Sigma}\,|^{-1/2}$ $\exp(-\mathbf{z}^T\boldsymbol{\Sigma}^{-1}\mathbf{z}/2)$. We work with $T$-point time blocks denoted $\mathbf{x}_{1:T} = \{\mathbf{x}_t\}_{t=1}^T$. The $i$th coordinate of $\mathbf{x}_t$ is $x_t^i$. For a function $f$, $\langle f(\mathbf{x}_{1:T}) \rangle$ denotes averaging over an ensemble of $\mathbf{x}_{1:T}$ blocks.

## 2 Zero Noise

The MOG source model employed in IFA [1] has the advantages that (i) it is capable of approximating arbitrary densities, and (ii) it can be learned efficiently from data by EM. The Gaussians correspond to the hidden states of the sources, labeled by $s$. Assume that at time $t$, source $i$ is in state $s_t^i = s$. Its signal $x_t^i$ is then generated by sampling from a Gaussian distribution with mean $\mu_s^i$ and variance $\nu_s^i$. In order to capture temporal statistics of the data, we endow the sources with temporal structure by introducing a transition matrix $a_{s's}^i$ between the states. Focusing on

a time block $t = 1, ..., T$, the resulting probabilistic model is defined by

$$p(s_t^i = s \mid s_{t-1}^i = s') = a_{s's}^i \, , \qquad p(s_0^i = s) = \pi_s^i \, ,$$

$$p(x_t^i \mid s_t^i = s) = \mathcal{G}(x_t^i - \mu_s^i, \nu_s^i) \, , \qquad p(\mathbf{y}_{1:T}) = \mid \det \mathbf{G} \mid^T p(\mathbf{x}_{1:T}) \, , \qquad (2)$$

where $p(\mathbf{x}_{1:T})$ is the joint density of all sources $x_t^i, i = 1, ..., L$ at all time points, and the last equation follows from $\mathbf{x}_t = \mathbf{G}\mathbf{y}_t$ with $\mathbf{G} = \mathbf{H}^{-1}$ being the unmixing matrix. As usual in the noise-free scenario (see [2]; section 7 of [1]), we are assuming that the mixing matrix is square and invertible.

The graphical model for the observed density $p(\mathbf{y}_{1:T} \mid W)$ defined by (2) is parametrized by $W = \{G_{ij}, \mu_s^i, \nu_s^i, \pi_s^i, a_{s's}^i\}$. This model describes each source as a first-order HMM; it reduces to a time-independent model if $a_{s's}^i = \pi_s^i$. Whereas temporal structure can be described by other means, e.g. a moving-average [4] or autoregressive [6] model, the HMM is advantageous since it models high-order temporal statistics and facilitates EM learning. Omitting the derivation, maximization with respect to $G_{ij}$ results in the incremental update rule

$$\delta \mathbf{G} = \epsilon \mathbf{G} - \epsilon \frac{1}{T} \sum_{t=1}^{T} \phi(\mathbf{x}_t) \mathbf{x}_t^T \mathbf{G} \, , \qquad (3)$$

where $\phi(x_t^i) = \sum_s \gamma_t^i(s)(x_t^i - \mu_s^i)/\nu_s^i$, and the natural gradient [3] was used; $\epsilon$ is an appropriately chosen learning rate. For the source parameters we obtain the update rules

$$\mu_s^i = \frac{\sum_t \gamma_t^i(s) x_t^i}{\sum_t \gamma_t^i(s)} \, , \qquad \nu_s^i = \frac{\sum_t \gamma_t^i(s)(x_t^i - \mu_s^i)^2}{\sum_t \gamma_t^i(s)} \, , \qquad a_{s's}^i = \frac{\sum_t \xi_t^i(s', s)}{\sum_t \gamma_{t-1}^i(s')} \, , \qquad (4)$$

with the initial probabilities updated via $\pi_s^i = \gamma_0^i(s)$. We used the standard HMM notation $\gamma_t^i(s) = p(s_t^i = s \mid x_{1:T}^i)$, $\xi_t^i(s', s) = p(s_{t-1}^i = s', s_t^i = s \mid x_{1:T}^i)$. These posterior densities are computed in the E-step for each source, which is given in terms of the data via $x_t^i = \sum_j G_{ij} y_t^j$, using the forward-backward procedure [7].

The algorithm (3–4) may be used in several possible generalized EM schemes. An efficient one is given by the following two-phase procedure: (i) freeze the source parameters and learn the separating matrix $\mathbf{G}$ using (3); (ii) freeze $\mathbf{G}$ and learn the source parameters using (4), then go back to (i) and repeat. Notice that the rule (3) is similar to a natural gradient version of Bell and Sejnowski's ICA rule [2]; in fact, the two coincide for time-independent sources where $\phi(x_i) = -\partial \log p(x_i)/\partial x_i$. We also recognize (4) as the Baum-Welch method. Hence, in phase (i) our algorithm separates the sources using a generalized ICA rule, whereas in phase (ii) it learns an HMM for each source.

**Remark.** Often one would like to model a given $L'$-variable time series in terms of a smaller number $L \leq L'$ of factors. In the framework of our noise-free model $\mathbf{y}_t = \mathbf{H}\mathbf{x}_t$, this can be achieved by applying the above algorithm to the $L$ largest principal components of the data; notice that if the data were indeed generated by $L$ factors, the remaining $L' - L$ principal components would vanish. Equivalently, one may apply the algorithm to the data directly, using a non-square $L \times L'$ unmixing matrix $\mathbf{G}$.

**Results.** Figure 1 demonstrates the performance of the above method on a $4 \times 4$ mixture of speech signals, which were passed through a non-linear function to modify their distributions. This mixture is inseparable to ICA because the source model used by the latter does not fit the actual source densities (see discussion in [1]). We also applied our dynamic network to a mixture of speech signals whose distributions

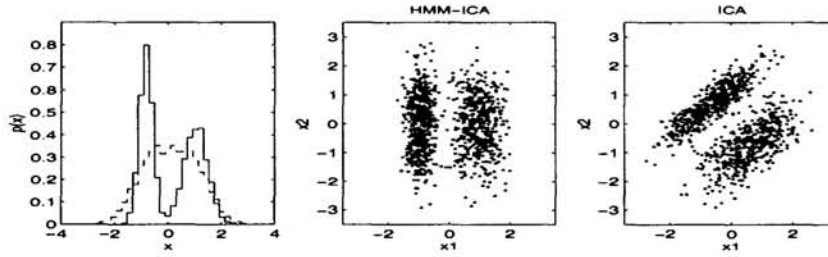

Figure 1: Left: Two of the four source distributions. Middle: Outputs of the EM algorithm (3–4) are nearly independent. Right: the outputs of ICA [2] are correlated.

were made Gaussian by an appropriate non-linear transformation. Since temporal information is crucial for separation in this case (see [4],[6]), this mixture is inseparable to ICA and IFA; however, the algorithm (3–4) accomplished separation successfully.

## 3  Isotropic Noise

We now turn to the case of non-zero noise $\mathbf{u}_t \neq 0$. We assume that the noise is white and has a zero-mean Gaussian distribution with covariance matrix $\Lambda$. In general, this case is computationally intractable (see section 4). The reason is that the E-step requires computing the posterior distribution $p(\mathbf{s}_{0:T}, \mathbf{x}_{1:T} \mid \mathbf{y}_{1:T})$ not only over the source states (as in the zero-noise case) but also over the source signals, and this posterior has a quite complicated structure. We now show that if we assume isotropic noise, i.e. $\Lambda_{ij} = \lambda \delta_{ij}$, as well as square invertible mixing as above, this posterior simplifies considerably, making learning and inference tractable. This is done by adapting an idea suggested in [8] to our dynamic probabilistic network.

We start by pre-processing the data using a linear transformation that makes their covariance matrix unity, i.e., $\langle \mathbf{y}_t \mathbf{y}_t^T \rangle = \mathbf{I}$ ('sphering'). Here $\langle \cdot \rangle$ denotes averaging over $T$-point time blocks. From (1) it follows that $\mathbf{H}\mathbf{S}\mathbf{H}^T = \lambda'\mathbf{I}$, where $\mathbf{S} = \langle \mathbf{x}_t \mathbf{x}_t^T \rangle$ is the diagonal covariance matrix of the sources, and $\lambda' = 1 - \lambda$. This, for a square invertible $\mathbf{H}$, implies that $\mathbf{H}^T\mathbf{H}$ is diagonal. In fact, since the unobserved sources can be determined only to within a scaling factor, we can set the variance of each source to unity and obtain the *orthogonality property* $\mathbf{H}^T\mathbf{H} = \lambda'\mathbf{I}$. It can be shown that the source posterior now factorizes into a product over the individual sources, $p(\mathbf{s}_{0:T}, \mathbf{x}_{1:T} \mid \mathbf{y}_{1:T}) = \prod_i p(s_{0:T}^i, x_{1:T}^i \mid \mathbf{y}_{1:T})$, where

$$p(s_{0:T}^i, x_{1:T}^i \mid \mathbf{y}_{1:T}) \propto \left[ \prod_{t=1}^T \mathcal{G}(x_t^i - \eta_t^i, \sigma_t^i) \cdot v_t^i p(s_t^i \mid s_{t-1}^i) \right] v_0^i p(s_0^i) . \qquad (5)$$

The means and variances at time $t$ in (5), as well as the quantities $v_t^i$, depend on both the data $\mathbf{y}_t$ and the states $s_t^i$; in particular, $\eta_t^i = (\sum_j H_{ji} y_t^j + \lambda \mu_s^i)/(\lambda' \nu_s + \lambda)$ and $\sigma_t^i = \lambda \nu_s^i/(\lambda' \nu_s + \lambda)$, using $s = s_t^i$; the expression for the $v_t^i$ are omitted. The transition probabilities are the same as in (2). Hence, the posterior distribution (5) effectively defines a new HMM for each source, with $\mathbf{y}_t$-dependent emission and transition probabilities.

To derive the learning rule for $\mathbf{H}$, we should first compute the conditional mean $\bar{\mathbf{x}}_t$ of the source signals at time $t$ given the data. This can be done recursively using (5) as in the forward-backward procedure. We then obtain

$$\mathbf{H} = \sqrt{\lambda'} \mathbf{C}(\mathbf{C}^T\mathbf{C})^{-1/2} , \qquad \mathbf{C} = \frac{1}{T} \sum_{t=1}^T \mathbf{y}_t \bar{\mathbf{x}}_t^T . \qquad (6)$$

This fractional form results from imposing the orthogonality constraint $\mathbf{H}^T\mathbf{H} = \lambda'\mathbf{I}$ using Lagrange multipliers and can be computed via a diagonalization procedure. The source parameters are computed using a learning rule (omitted) similar to the noise-free rule (4). It is easy to derive a learning rule for the noise level $\lambda$ as well; in fact, the ordinary FA rule would suffice. We point out that, while this algorithm has been derived for the case $L = L'$, it is perfectly well defined (though sub-optimal: see below) for $L \leq L'$.

## 4  Non-Isotropic Noise

The general case of non-isotropic noise and non-square mixing is computationally intractable. This is because the exact E-step requires summing over all possible source configurations $(s_{t_1}^1, ..., s_{t_L}^L)$ at all times $t_1, ..., t_L = 1, ..., T$. The intractability problem stems from the fact that, while the sources are independent, the sources *conditioned on a data vector* $\mathbf{y}_{1:T}$ are correlated, resulting in a large number of hidden configurations. This problem does not arise in the noise-free case, and can be avoided in the case of isotropic noise and square mixing using the orthogonality property; in both cases, the exact posterior over the sources factorizes.

The EM algorithm derived below is based on a variational approach. This approach was introduced in [9] in the context of sigmoid belief networks, but constitutes a general framework for ML learning in intractable probabilistic networks; it was used in a HMM context in [10]. The idea is to use an approximate but tractable posterior to place a lower bound on the likelihood, and optimize the parameters by maximizing this bound.

A starting point for deriving a bound on the likelihood $\mathcal{L}$ is Neal and Hinton's [11] formulation of the EM algorithm:

$$\mathcal{L} = \log p(\mathbf{y}_{1:T}) \geq \sum_{t=1}^{T} E_q \log p(\mathbf{y}_t \mid \mathbf{x}_t) + \sum_{i=1}^{L} E_q \log p(s_{0:T}^i, x_{1:T}^i) - E_q \log q \,, \quad (7)$$

where $E_q$ denotes averaging with respect to an arbitrary posterior density over the hidden variables given the observed data, $q = q(\mathbf{s}_{0:T}, \mathbf{x}_{1:T} \mid \mathbf{y}_{1:T})$. Exact EM, as shown in [11], is obtained by maximizing the bound (7) with respect to both the posterior $q$ (corresponding to the E-step) and the model parameters $W$ (M-step). However, the resulting $q$ is the true but intractable posterior. In contrast, in variational EM we choose a $q$ that differs from the true posterior, but facilitates a tractable E-step.

**E-Step.**  We use $q(\mathbf{s}_{0:T}, \mathbf{x}_{1:T} \mid \mathbf{y}_{1:T}) = \prod_i q(s_{0:T}^i \mid \mathbf{y}_{1:T}) \prod_t q(\mathbf{x}_t \mid \mathbf{y}_{1:T})$, parametrized as

$$q(s_t^i = s \mid s_{t-1}^i = s', \mathbf{y}_{1:T}) \propto \lambda_{s,t}^i a_{s's}^i \,, \quad q(s_0^i = s \mid \mathbf{y}_{1:T}) \propto \lambda_{s,t}^i \pi_s^i \,,$$
$$q(\mathbf{x}_t \mid \mathbf{y}_{1:T}) = \mathcal{G}(\mathbf{x}_t - \boldsymbol{\rho}_t, \boldsymbol{\Sigma}_t) \,. \quad (8)$$

Thus, the variational transition probabilities in (8) are described by multiplying the original ones $a_{s's}^i$ by the parameters $\lambda_{s,t}^i$, subject to the normalization constraints. The source signals $\mathbf{x}_t$ at time $t$ are jointly Gaussian with mean $\boldsymbol{\rho}_t$ and covariance $\boldsymbol{\Sigma}_t$. The means, covariances and transition probabilities are all time- and data-dependent, i.e., $\boldsymbol{\rho}_t = f(\mathbf{y}_{1:T}, t)$ etc. This parametrization scheme is motivated by the form of the posterior in (5); notice that the quantities $\eta_t^i, \sigma_t^i, v_{s,t}^i$ there become the *variational parameters* $\rho_t^i, \Sigma_t^{ij}, \lambda_{s,t}^i$ of (8). A related scheme was used in [10] in a different context. Since these parameters will be adapted independently of the model parameters, the non-isotropic algorithm is expected to give superior results compared to the isotropic one.

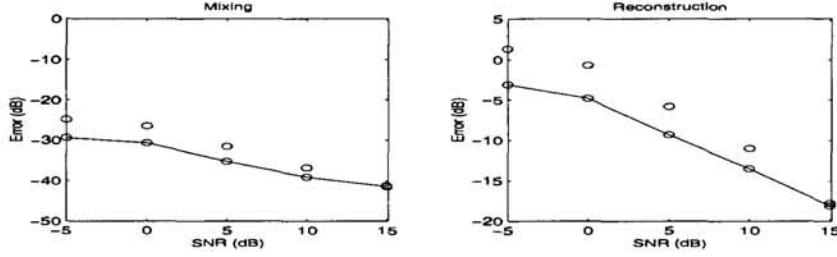

Figure 2: Left: quality of the model parameter estimates. Right: quality of the source reconstructions. (See text).

Of course, in the *true* posterior the $\mathbf{x}_t$ are correlated, both temporally among themselves and with $\mathbf{s}_t$, and the latter do not factorize. To best approximate it, the variational parameters $V = \{\rho_t^i, \Sigma_t^{ij}, \lambda_{s,t}^i\}$ are optimized to maximize the bound on $\mathcal{L}$, or equivalently to minimize the KL distance between $q$ and the true posterior. This requirement leads to the fixed point equations

$$\rho_t = (\mathbf{H}^T \mathbf{\Lambda}^{-1} \mathbf{H} + \mathbf{B}_t)^{-1}(\mathbf{H}^T \mathbf{\Lambda}^{-1} \mathbf{y}_t + \mathbf{b}_t) , \quad \Sigma_t = (\mathbf{H}^T \mathbf{\Lambda}^{-1} \mathbf{H} + \mathbf{B}_t)^{-1} ,$$

$$\lambda_{s,t}^i = \frac{1}{z_t^i} \exp\left[ -\frac{1}{2}\log \nu_s^i - \frac{(\rho_t^i - \mu_s^i)^2 + \Sigma_t^{ii}}{2\nu_s^i} \right] , \tag{9}$$

where $B_t^{ij} = \sum_s [\gamma_t^i(s)/\nu_s^i]\delta_{ij}$, $b_t^i = \sum_s \gamma_t^i(s)\mu_s^i/\nu_s^i$, and the factors $z_t^i$ ensure normalization. The HMM quantities $\gamma_t^i(s)$ are computed by the forward-backward procedure using the *variational* transition probabilities (8). The variational parameters are determined by solving eqs. (9) iteratively for each block $\mathbf{y}_{1:T}$; in practice, we found that less then 20 iterations are usually required for convergence.

**M-Step.** The update rules for $W$ are given for the mixing parameters by

$$\mathbf{H} = \left[\sum_t \mathbf{y}_t \rho_t^T\right] \left[\sum_t (\rho_t \rho_t^T + \Sigma_t)\right]^{-1} , \quad \mathbf{\Lambda} = \frac{1}{T}\sum_t (\mathbf{y}_t \mathbf{y}_t^T - \mathbf{y}_t \rho_t^T \mathbf{H}^T) , \tag{10}$$

and for the source parameters by

$$\mu_s^i = \frac{\sum_t \gamma_t^i(s)\rho_t^i}{\sum_t \gamma_t^i(s)} , \quad \nu_s^i = \frac{\sum_t \gamma_t^i(s)((\rho_t^i - \mu_s^i)^2 + \Sigma_t^{ii})}{\sum_t \gamma_t^i(s)} ,$$

$$a_{s's}^i = \frac{\sum_t \xi_t^i(s', s)}{\sum_t \gamma_{t-1}^i(s')} , \quad \pi_s^i = \gamma_0^i(s) , \tag{11}$$

where the $\xi_t^i(s', s)$ are computed using the variational transition probabilities (8). Notice that the learning rules for the source parameters have the Baum-Welch form, in spite of the correlations between the conditioned sources. In our variational approach, these correlations are hidden in $V$, as manifested by the fact that the fixed point equations (9) couple the parameters $V$ across time points (since $\gamma_t^i(s)$ depends on $\lambda_{s,t=1:T}^i$) and sources.

**Source Reconstruction.** From $q(\mathbf{x}_t \mid \mathbf{y}_{1:T})$ (8), we observe that the MAP source estimate is given by $\hat{\mathbf{x}}_t = \rho_t(\mathbf{y}_{1:T})$, and depends on both $W$ and $V$.

**Results.** The above algorithm is demonstrated on a source separation task in Figure 2. We used 6 speech signals, transformed by non-linearities to have arbitrary one-point densities, and mixed by a random $8 \times 6$ matrix $\mathbf{H}_0$. Different signal-to-noise (SNR) levels were used. The error in the estimated $\mathbf{H}$ (left, solid line) is quantified by the size of the non-diagonal elements of $(\mathbf{H}^T \mathbf{H})^{-1}\mathbf{H}^T \mathbf{H}_0$ relative to the

diagonal; the results obtained by IFA [1], which does not use temporal information, are plotted for reference (dotted line). The mean squared error of the reconstructed sources (right, solid line) and the corresponding IFA result (right, dashed line) are also shown. The estimate and reconstruction errors of this algorithm are consistently smaller than those of IFA, reflecting the advantage of exploiting the temporal structure of the data. Additional experiments with different numbers of sources and sensors gave similar results. Notice that this algorithm, unlike the previous two, allows both $L \leq L'$ and $L > L'$. We also considered situations where the number of sensors was smaller than the number of sources; the separation quality was good, although, as expected, less so than in the opposite case.

## 5  Conclusion

An important issue that has not been addressed here is model selection. When applying our algorithms to an arbitrary dataset, the number of factors and of HMM states for each factor should be determined. Whereas this could be done, in principle, using cross-validation, the required computational effort would be fairly large. However, in a recent paper [12] we develop a new framework for Bayesian model selection, as well as model averaging, in probabilistic networks. This framework, termed *Variational Bayes*, proposes an EM-like algorithm which approximates full posterior distributions over not only hidden variables but also parameters and model structure, as well as predictive quantities, in an analytical manner. It is currently being applied to the algorithms presented here with good preliminary results.

One field in which our approach may find important applications is speech technology, where it suggests building more economical signal models based on combining independent low-dimensional HMMs, rather than fitting a single complex HMM. It may also contribute toward improving recognition performance in noisy, multi-speaker, reverberant conditions which characterize real-world auditory scenes.

## References

[1] Attias, H. (1999). Independent factor analysis. *Neur. Comp.* **11**, 803-851.
[2] Bell, A.J. & Sejnowski, T.J. (1995). An information-maximization approach to blind separation and blind deconvolution. *Neur. Comp.* **7**, 1129-1159.
[3] Amari, S., Cichocki, A. & Yang, H.H. (1996). A new learning algorithm for blind signal separation. *Adv. Neur. Info. Proc. Sys.* **8**, 757-763 (Ed. by Touretzky, D.S. et al). MIT Press, Cambridge, MA.
[4] Pearlmutter, B.A. & Parra, L.C. (1997). Maximum likelihood blind source separation: A context-sensitive generalization of ICA. *Adv. Neur. Info. Proc. Sys.* **9**, 613-619 (Ed. by Mozer, M.C. et al). MIT Press, Cambridge, MA.
[5] Hyvärinen, A. & Oja, E. (1997). A fast fixed-point algorithm for independent component analysis. *Neur. Comp.* **9**, 1483-1492.
[6] Attias, H. & Schreiner, C.E. (1998). Blind source separation and deconvolution: the dynamic component analysis algorithm. *Neur. Comp.* **10**, 1373-1424.
[7] Rabiner, L. & Juang, B.-H. (1993). *Fundamentals of Speech Recognition.* Prentice Hall, Englewood Cliffs, NJ.
[8] Lee, D.D. & Sompolinsky, H. (1999), unpublished; D.D. Lee, personal communication.
[9] Saul, L.K., Jaakkola, T., and Jordan, M.I. (1996). Mean field theory of sigmoid belief networks. *J. Art. Int. Res.* **4**, 61-76.
[10] Ghahramani, Z. & Jordan, M.I. (1997). Factorial hidden Markov models. *Mach. Learn.* **29**, 245-273.
[11] Neal, R.M. & Hinton, G.E. (1998). A view of the EM algorithm that justifies incremental, sparse, and other variants. *Learning in Graphical Models*, 355-368 (Ed. by Jordan, M.I.). Kluwer Academic Press.
[12] Attias, H. (2000). A variational Bayesian framework for graphical models. *Adv. Neur. Info. Proc. Sys.* **12** (Ed. by Leen, T. et al). MIT Press, Cambridge, MA.